# RCC Cannot Compute Certain FSA, Even with Arbitrary Transfer Functions

**Mark Ring**
RWCP Theoretical Foundation GMD Laboratory
GMD — German National Research Center for Information Technology
Schloss Birlinghoven
D-53 754 Sankt Augustin, Germany
email: Mark.Ring@GMD.de

## Abstract

Existing proofs demonstrating the computational limitations of Recurrent Cascade Correlation and similar networks (Fahlman, 1991; Bachrach, 1988; Mozer, 1988) explicitly limit their results to units having sigmoidal or hard-threshold transfer functions (Giles et al., 1995; and Kremer, 1996). The proof given here shows that *for any* finite, discrete transfer function used by the units of an RCC network, there are finite-state automata (FSA) that the network cannot model, no matter how many units are used. The proof also applies to *continuous* transfer functions with a finite number of fixed-points, such as sigmoid and radial-basis functions.

## 1  Introduction

The Recurrent Cascade Correlation (RCC) network was proposed by Fahlman (1991) to offer a fast and efficient alternative to fully connected recurrent networks. The network is arranged such that each unit has only a single recurrent connection: the connection that goes from itself to itself. Networks with the same structure have been proposed by Mozer (Mozer, 1988) and Bachrach (Bachrach, 1988). This structure is intended to allow simplified training of recurrent networks in the hopes of making them computationally feasible. However, this increase in efficiency comes at the cost of computational power: the networks' computational capabilities are limited *regardless of the power of their activation functions*. The remaining input to each unit consists of the input to the network as a whole together with the outputs from all units lower in the RCC network. Since it is the structure of the network and not the learning algorithm that is of interest here, only the structure will be described in detail.

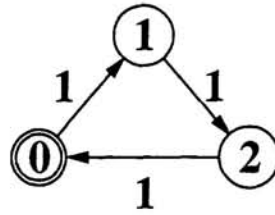

Figure 1: This finite-state automaton was shown by Giles et al. (1995) to be unrepresentable by an RCC network whose units have hard-threshold or sigmoidal transfer functions. The arcs are labeled with transition labels of the FSA which are given as input to the RCC network. The nodes are labeled with the output values that the network is required to generate. The node with an inner circle is an accepting or *halting* state.

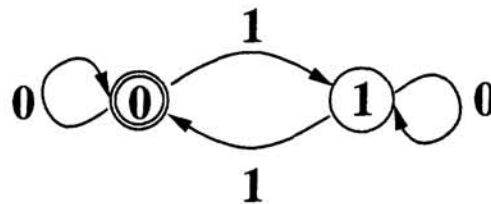

Figure 2: This finite-state automaton is one of those shown by Kremer (1996) not to be representable by an RCC network whose units have a hard-threshold or sigmoidal transfer function. This FSA computes the parity of the inputs seen so far.

The functionality of a network of $N$ RCC units, $U_0, ..., U_{N-1}$ can be described in the following way:

$$V_0(t) = f_0(\vec{i}(t), V_0(t-1)) \tag{1}$$

$$V_x(t) = f_x(\vec{i}(t), V_x(t-1), V_{x-1}(t), V_{x-2}(t), ..., V_0(t)), \tag{2}$$

where $V_x(t)$ is the output value of $U_x$ at time step $t$, and $\vec{i}(t)$ is the input to the network at time step $t$. The value of each unit is determined from: (1) the network input at the current time step, (2) its own value at the previous time step, and (3) the output values of the units lower in the network at the current time step. Since learning is not being considered here, the weights are assumed to be constant.

## 2   Existing Proofs

The proof of Giles, et al (1995) showed that an RCC network whose units had a hard-threshold or sigmoidal transfer function cannot produce outputs that oscillate with a period greater than two when the network input is constant. (An oscillation has a period of $x$ if it repeats itself every $x$ steps.) Thus, the FSA shown in Figure 1 cannot be modeled by such an RCC network, since its output (shown as node labels) oscillates at a period greater than two given constant input. Kremer (1996) refined the class of FSA representable by an RCC network showing that, if the input to the net oscillates with period $p$, then the output can only oscillate with a period of $\omega$, where $\omega$ is one of $p$'s factors (or of $2p$'s factors if $p$ is odd). An unrepresentable example, therefore, is the parity FSA shown in Figure 2, whose output has a period of four given the following input (of period two): $0, 1, 0, 1, ....$

Both proofs, that by Giles et al. and that by Kremer, are explicitly designed with

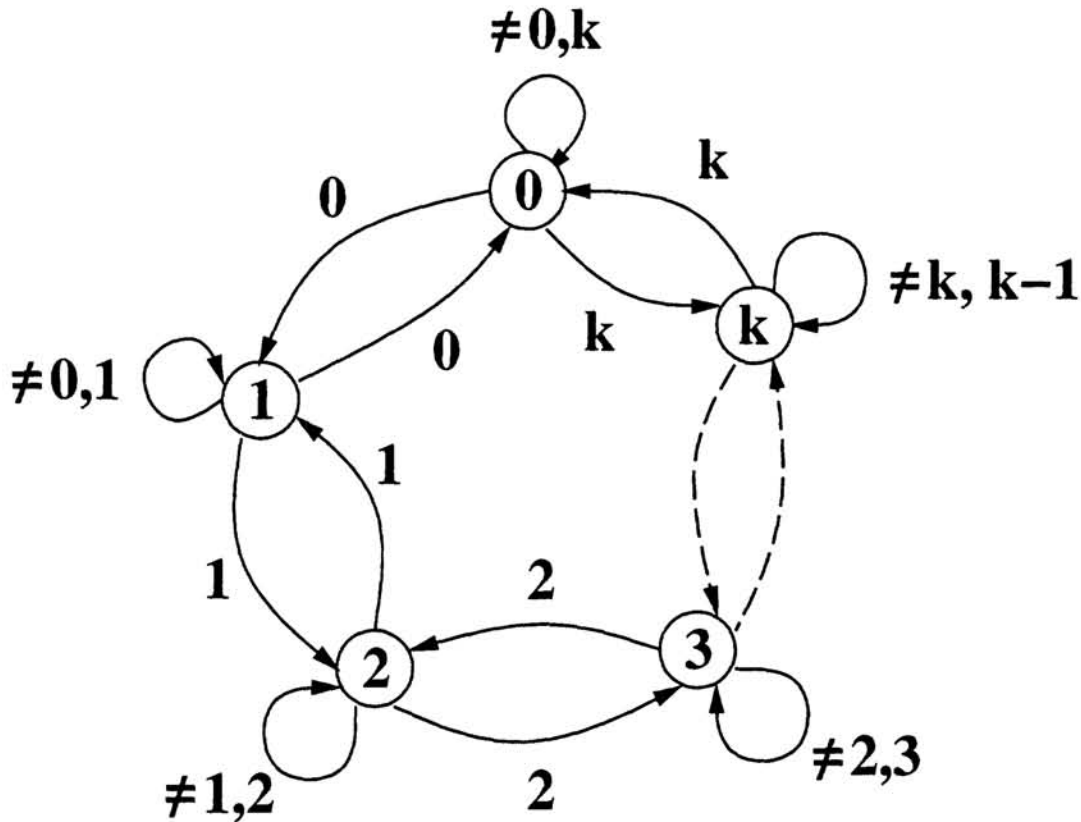

Figure 3: This finite-state automaton cannot be modeled with any RCC network whose units are capable of representing only $k$ discrete outputs. The values within the circles are the state names and the output expected from the network. The arcs describe transitions from state to state, and their values represent the input given to the network when the transition is made. The dashed lines indicate an arbitrary number of further states between state 3 and state $k$ which are connected in the same manner as states 1, 2, and 3. (All states are halting states.)

hard-threshold and sigmoidal transfer functions in mind, and can say nothing about other transfer functions. In other words, these proofs do not demonstrate the limitations of the RCC-type network *structure*, but about the use of threshold units within this structure. The following proof is the first that actually demonstrates the limitations of the single-recurrent-link network structure.

## 3  Details of the Proof

This section proves that RCC networks are incapable even in principle of modeling certain kinds of FSA, regardless of the sophistication of each unit's transfer function, provided only that the transfer function be discrete and finite, meaning only that the units of the RCC network are capable of generating a fixed number, $k$, of distinct output values. (Since all functions implemented on a discrete computer fall into this category, this assumption is minor. Furthermore, as will be discussed in Section 4, the outputs of most interesting continuous transfer functions reduce to only a small number of distinct values.) This generalized RCC network is proven here to be incapable of modeling the finite-state automaton shown in Figure 3.

For ease of exposition, let us call any FSA of the form shown in Figure 3 an $RF^{k+1}$ for *Ring FSA with $k + 1$ states*.[1] Further, call a unit whose output can be any of $k$ distinct values and whose input includes its own previous output, a $DRU^k$ for *Discrete Recurrent Unit*. These units are a generalization of the units used by RCC networks in that the specific transfer function is left unspecified. By proving the network is limited when its units are $DRU^k$'s proves the limitations of the network's *structure* regardless of the transfer function used.

Clearly, a $DRU^{k+1}$ with a sufficiently sophisticated transfer function could by itself model an $RF^{k+1}$ by simply allocating one of its $k + 1$ output values for each of the $k + 1$ states. At each step it would receive as input the last state of the FSA and the next transition and could therefore compute the next state. By restricting the units in the least conceivable manner, i.e., by reducing the number of distinct output values to $k$, the RCC network becomes incapable of modeling any $RF^{k+1}$ regardless of how many $DRU^k$'s the network contains. This will now be proven.

The proof is inductive and begins with the first unit in the network, which, after being given certain sequences of inputs, becomes incapable of distinguishing among any states of the FSA. The second step, the inductive step, proves that no finite number of such units can assist a unit higher in the RCC network in making a distinction between any states of the $RF^{k+1}$.

**Lemma 1** *No $DRU^k$ whose input is the current transition of an $RF^{k+1}$ can reliably distinguish among any states of the $RF^{k+1}$. More specifically, at least one of the $DRU^k$'s $k$ output values can be generated in all of the $RF^{k+1}$'s $k + 1$ states.*

**Proof:** Let us name the $DRU^k$'s $k$ distinct output values $V^0, V^1, ..., V^{k-1}$. The mapping function implemented by the $DRU^k$ can be expressed as follows:

$$(V^x, i) \Rightarrow V^y,$$

which indicates that when the unit's last output was $V^x$ and its current input is $i$, then its next output is $V^y$.

Since an $RF^k$ is cyclical, the arithmetic in the following will also be cyclical (i.e., modular):

$$x \oplus y \equiv \begin{cases} x + y & \text{if } x + y < k \\ x + y - k & \text{if } x + y \geq k \end{cases}$$

$$x \ominus y \equiv \begin{cases} x - y & \text{if } x \geq y \\ x + k - y & \text{if } x < y \end{cases}$$

where $0 \leq x < k$ and $0 \leq y < k$.

Since it is impossible for the $DRU^k$ to represent each of the $RF^{k+1}$'s $k+1$ states with a distinct output value, at least two of these states must be represented ambiguously by the same value. That is, there are two $RF^{k+1}$ states $a$ and $b$ and one $DRU^k$ value $V^{a/b}$ such that $V^{a/b}$ can be generated by the unit both when the FSA is in state $a$ and when it is in state $b$. Furthermore, this value *will* be generated by the unit given an appropriate sequence of inputs. (Otherwise the value is *unreachable*, serves no purpose, and can be discarded, reducing the unit to a $DRU^{k-1}$.)

Once the $DRU^k$ has generated $V^{a/b}$, it cannot in the next step distinguish whether the FSA's current state is $a$ or $b$. Since the FSA could be in either state $a$ or $b$, the next state after a $b$ transition could be either $a$ or $b \oplus 1$. That is:

$$(V^{a/b}, b) \Rightarrow V^{a/b \oplus 1}, \tag{3}$$

where $a \ominus b \geq b \ominus a$ and $k > 1$. This new output value $V^{a/b \oplus 1}$ can therefore be generated when the FSA is in either state $a$ or state $b \oplus 1$. By repeatedly replacing $b$ with $b \oplus 1$ in Equation 3, all states from $b$ to $a \ominus 1$ can be shown to share output values with state $a$, i.e., $V^{a/b}$, $V^{a/b \oplus 1}$, $V^{a/b \oplus 2}$, ..., $V^{a/a \ominus 2}$, $V^{a/a \ominus 1}$ all exist.

Repeatedly substituting $a \ominus 1$ and $a$ for $a$ and $b$ respectively in the last paragraph produces values $V^{x/y}$ $\forall x, y \in 0, 1, ..., k+1$. There is, therefore, at least one value that can be generated by the unit in both states of every possible pair of states.

Since there are $\binom{k+1}{2}$ distinct pairs but only $k$ distinct output values, and since

$$\left\lceil \frac{\binom{k+1}{2}}{k} \right\rceil > 1,$$

when $k > 1$, then not all of these pairs can be represented by unique $V$ values. At least two of these pairs must share the same output value, and this implies that some $V^{a/b/c}$ exists that can be output by the unit in any of the three FSA states $a, b,$ and $c$.

Starting with

$$(V^{a/b/c}, c) \Rightarrow V^{a/b/c \oplus 1},$$

and following the same argument given above for $V^{a/b}$, there must be a $V^{x/y/z}$ for all triples of states $x, y,$ and $z$. Since there are $\binom{k+1}{3}$ distinct triples but only $k$ distinct output values, and since

$$\left\lceil \frac{\binom{k+1}{3}}{k} \right\rceil > 1,$$

where $k > 3$, some $V^{a/b/c/d}$ must also exist.

This argument can be followed repeatedly since:

$$\left\lceil \frac{\binom{k+1}{m}}{k} \right\rceil > 1,$$

for all $m < k + 1$, including when $m = k$. Therefore, there is at least one $V^{0/1/2/.../k/k+1}$ that can be output by the unit in all $k + 1$ states of the $RF^{k+1}$. Call this value and any other that can be generated in all FSA states $V^k$. All $V^k$'s are reachable (else they could be discarded and the above proof applied for $DRU^l, l < k$). When a $V^k$ is output by a $DRU^k$, it does not distinguish any states of the $RF^{k+1}$.

**Lemma 2** *Once a $DRU^k$ outputs a $V^k$, all future outputs will also be $V^k$'s.*

**Proof:** The proof is simply by inspection, and is shown in the following table:

| Actual State | Transition | Next State |
|:---:|:---:|:---:|
| $x$ | $x$ | $x \oplus 1$ |
| $x \oplus 1$ | $x$ | $x$ |
| $x \oplus 2$ | $x$ | $x \oplus 2$ |
| $x \oplus 3$ | $x$ | $x \oplus 3$ |
| ... | ... | ... |
| $x \ominus 2$ | $x$ | $x \ominus 2$ |
| $x \ominus 1$ | $x$ | $x \ominus 1$ |

If the unit's last output value was a $V^k$, then the FSA might be in any of its $k+1$ possible states. As can be seen, if at this point any of the possible transitions is given as input, the next state can also be any of the $k+1$ possible states. Therefore, no future input can ever serve to lessen the unit's ambiguity.

**Theorem 1** *An RCC network composed of any finite number of $DRU^k$'s cannot model an $RF^{k+1}$.*

**Proof:** Let us describe the transitions of an RCC network of N units by using the following notation:

$$(\langle V_{N-1}, V_{N-2}, ..., V_1, V_0 \rangle, i) \Rightarrow \langle V'_{N-1}, V'_{N-2}, ..., V'_1, V'_0 \rangle,$$

where $V_m$ is the output value of the $m$'th unit (i.e., $U_m$) before the given input, $i$, is seen by the network, and $V'_m$ is $U_m$'s value after $i$ has been processed by the network. The first unit, $U_0$, receives only $i$ and $V_0$ as input. Every other unit $U_x$ receives as input $i$ and $V_x$ as well as $V'_y, y < x$.

Lemma 1 shows that the first unit, $U_0$, will eventually generate a value $V_0^k$, which can be generated in any of the $RF^{k+1}$ states. From Lemma 2, the unit will continue to produce $V_0^k$ values after this point.

Given any finite number $N$ of $DRU^k$'s, $U_{m-1}, ..., U_0$ that are producing their $V_k$ values, $V_{N-1}^k, ..., V_0^k$, the next higher unit, $U_N$, will be incapable of disambiguating all states by itself, i.e., at least two FSA states, $a$ and $b$, will have overlapping output values, $V_N^{a/b}$. Since none of the units $U_{N-1}, ..., U_0$ can distinguish between any states (including $a$ and $b$),

$$(\langle V_N^{a/b}, V_{N-1}^k, ..., V_1^k, V_0^k \rangle, b) \Rightarrow \langle V_N^{a/b \oplus 1}, V_{N-1}^k, ..., V_1^k, V_0^k \rangle,$$

assuming that $b \ominus a \geq a \ominus b$ and $k > 1$. The remainder of the proof follows identically along the lines developed for Lemmas 1 and 2. The result of this development is that $U_N$ also has a set of reachable output values $V_N^k$ that can be produced in any state of the FSA. Once one such value is produced, no less-ambiguous value is ever generated. Since no RCC network containing any number of $DRU^k$'s can over time distinguish among any states of an $RF^{k+1}$, no such RCC network can model such an FSA.

## 4   Continuous Transfer Functions

Sigmoid functions can generate a theoretically infinite number of output values; if represented with 32 bits, they can generate $2^{32}$ outputs. This hardly means, however, that all such values are of use. In fact, as was shown by Giles et al. (1995), if the input remains constant for a long enough period of time (as it can in all $RF^{k+1}$'s), the output of sigmoid units will converge to a constant value (a fixed point) or oscillate between two values. This means that a unit with a sigmoid transfer function is in principle a $DRU^2$. Most useful continuous transfer functions (radial-basis functions, for example), exhibit the same property, reducing to only a small number of distinct output values when given the same input repeatedly. The results shown here are therefore not merely theoretical, but are of real practical significance and apply to any network whose recurrent links are restricted to self connections.

## 5   Conclusion

No RCC network can model any FSA containing an $RF^{k+1}$ (such as that shown in Figure 3), given units limited to generating $k$ possible output values, regardless

of the sophistication of the transfer function that generates these values. This places an upper bound on the computational capabilities of an RCC network. Less sophisticated transfer functions, such as the sigmoid units investigated by Giles et al. and Kremer may have even greater limitations. Figure 2, for example, could be modeled by a single sufficiently sophisticated $DRU^2$, but cannot be modeled by an RCC network composed of hard-threshold or sigmoidal units (Giles et al., 1995; Kremer, 1996) because these units cannot exploit all mappings from inputs to outputs. By not assuming arbitrary transfer functions, previous proofs could not isolate the network's structure as the source of RCC's limitations.

## Footnotes

[1] Thanks to Mike Mozer for suggesting this catchy name.

# References

Bachrach, J. R. (1988). Learning to represent state. Master's thesis, Department of Computer and Information Sciences, University of Massachusetts, Amherst, MA 01003.

Fahlman, S. E. (1991). The recurrent cascade-correlation architecture. In Lippmann, R. P., Moody, J. E., and Touretzky, D. S., editors, *Advances in Neural Information Processing Systems 3*, pages 190–196, San Mateo, California. Morgan Kaufmann Publishers.

Giles, C., Chen, D., Sun, G., Chen, H., Lee, Y., and Goudreau, M. (1995). Constructive learning of recurrent neural networks: Problems with recurrent cascade correlation and a simple solution. *IEEE Transactions on Neural Networks*, 6(4):829.

Kremer, S. C. (1996). Finite state automata that recurrent cascade-correlation cannot represent. In Touretzky, D. S., Mozer, M. C., and Hasselno, M. E., editors, *Advances in Neural Information Processing Systems 8*, pages 679–686. MIT Press. In Press.

Mozer, M. C. (1988). A focused back-propagation algorithm for temporal pattern recognition. Technical Report CRG–TR–88–3, Department of Psychology, University of Toronto.
